# Fast Image Deconvolution
# using Hyper-Laplacian Priors

**Dilip Krishnan,**
Dept. of Computer Science,
Courant Institute,
New York University
dilip@cs.nyu.edu

**Rob Fergus,**
Dept. of Computer Science,
Courant Institute,
New York University
fergus@cs.nyu.edu

## Abstract

The heavy-tailed distribution of gradients in natural scenes have proven effective priors for a range of problems such as denoising, deblurring and super-resolution. These distributions are well modeled by a hyper-Laplacian $\left(p(x) \propto e^{-k|x|^{\alpha}}\right)$, typically with $0.5 \leq \alpha \leq 0.8$. However, the use of sparse distributions makes the problem non-convex and impractically slow to solve for multi-megapixel images. In this paper we describe a deconvolution approach that is several orders of magnitude faster than existing techniques that use hyper-Laplacian priors. We adopt an alternating minimization scheme where one of the two phases is a non-convex problem that is separable over pixels. This per-pixel sub-problem may be solved with a lookup table (LUT). Alternatively, for two specific values of $\alpha$, $1/2$ and $2/3$ an analytic solution can be found, by finding the roots of a cubic and quartic polynomial, respectively. Our approach (using either LUTs or analytic formulae) is able to deconvolve a 1 megapixel image in less than $\sim 3$ seconds, achieving comparable quality to existing methods such as iteratively reweighted least squares (IRLS) that take $\sim 20$ minutes. Furthermore, our method is quite general and can easily be extended to related image processing problems, beyond the deconvolution application demonstrated.

## 1 Introduction

Natural image statistics are a powerful tool in image processing, computer vision and computational photography. Denoising [14], deblurring [3], transparency separation [11] and super-resolution [20], are all tasks that are inherently ill-posed. Priors based on natural image statistics can regularize these problems to yield high-quality results. However, digital cameras now have sensors that record images with tens of megapixels (MP), e.g. the latest Canon DSLRs have over 20MP. Solving the above tasks for such images in a reasonable time frame (i.e. a few minutes or less), poses a severe challenge to existing algorithms. In this paper we focus on one particular problem: non-blind deconvolution, and propose an algorithm that is practical for very large images while still yielding high quality results.

Numerous deconvolution approaches exist, varying greatly in their speed and sophistication. Simple filtering operations are very fast but typically yield poor results. Most of the best-performing approaches solve globally for the corrected image, encouraging the marginal statistics of a set of filter outputs to match those of uncorrupted images, which act as a prior to regularize the problem. For these methods, a trade-off exists between accurately modeling the image statistics and being able to solve the ensuing optimization problem efficiently. If the marginal distributions are assumed to be Gaussian, a closed-form solution exists in the frequency domain and FFTs can be used to recover the image very quickly. However, real-world images typically have marginals that are non-Gaussian, as shown in Fig. 1, and thus the output is often of mediocre quality. A common approach is to assume the marginals have a Laplacian distribution. This allows a number of fast $\ell_1$ and related TV-norm methods [17, 22] to be deployed, which give good results in a reasonable time. However, studies

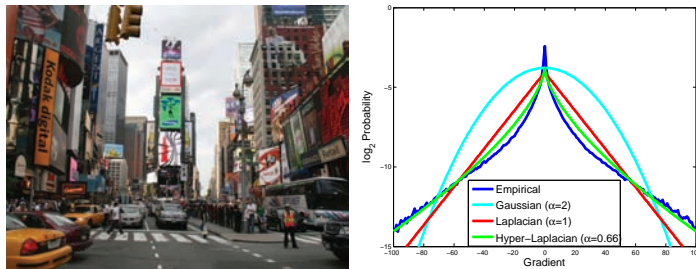

Figure 1: A hyper-Laplacian with exponent $\alpha = 2/3$ is a better model of image gradients than a Laplacian or a Gaussian. **Left:** A typical real-world scene. **Right:** The empirical distribution of gradients in the scene (blue), along with a Gaussian fit (cyan), a Laplacian fit (red) and a hyper-Laplacian with $\alpha = 2/3$ (green). Note that the hyper-Laplacian fits the empirical distribution closely, particularly in the tails.

of real-world images have shown the marginal distributions have significantly heavier tails than a Laplacian, being well modeled by a hyper-Laplacian [4, 10, 18]. Although such priors give the best quality results, they are typically far slower than methods that use either Gaussian or Laplacian priors. This is a direct consequence of the problem becoming non-convex for hyper-Laplacians with $\alpha < 1$, meaning that many of the fast $\ell_1$ or $\ell_2$ tricks are no longer applicable. Instead, standard optimization methods such as conjugate gradient (CG) must be used. One variant that works well in practice is iteratively reweighted least squares (IRLS) [19] that solves a series of weighted least-squares problems with CG, each one an $\ell_2$ approximation to the non-convex problem at the current point. In both cases, typically hundreds of CG iterations are needed, each involving an expensive convolution of the blur kernel with the current image estimate.

In this paper we introduce an efficient scheme for non-blind deconvolution of images using a hyper-Laplacian image prior for $0 < \alpha \le 1$. Our algorithm uses an alternating minimization scheme where the non-convex part of the problem is solved in one phase, followed by a quadratic phase which can be efficiently solved in the frequency domain using FFTs. We focus on the first phase where at each pixel we are required to solve a non-convex separable minimization. We present two approaches to solving this sub-problem. The first uses a lookup table (LUT); the second is an analytic approach specific to two values of $\alpha$. For $\alpha = 1/2$ the global minima can be determined by finding the roots of a cubic polynomial analytically. In the $\alpha = 2/3$ case, the polynomial is a quartic whose roots can also be found efficiently in closed-form. Both IRLS and our approach solve a series of approximations to the original problem. However, in our method each approximation is solved by alternating between the two phases above a few times, thus avoiding the expensive CG descent used by IRLS. This allows our scheme to operate several orders of magnitude faster. Although we focus on the problem of non-blind deconvolution, it would be straightforward to adapt our algorithm to other related problems, such as denoising or super-resolution.

## 1.1 Related Work

Hyper-Laplacian image priors have been used in a range of settings: super-resolution [20], transparency separation [11] and motion deblurring [9]. In work directly relevant to ours, Levin *et al.* [10] and Joshi *et al.* [7] have applied them to non-blind deconvolution problems using IRLS to solve for the deblurred image. Other types of sparse image prior include: Gaussian Scale Mixtures (GSM) [21], which have been used for image deblurring [3] and denoising [14] and student-T distributions for denoising [25, 16]. With the exception of [14], these methods use CG and thus are slow.

The alternating minimization that we adopt is a common technique, known as half-quadratic splitting, originally proposed by Geman and colleagues [5, 6]. Recently, Wang *et al.* [22] showed how it could be used with a total-variation (TV) norm to deconvolve images. Our approach is closely related to this work: we also use a half-quadratic minimization, but the per-pixel sub-problem is quite different. With the TV norm it can be solved with a straightforward shrinkage operation. In our work, as a consequence of using a sparse prior, the problem is non-convex and solving it efficiently is one of the main contributions of this paper.

Chartrand [1, 2] has introduced non-convex compressive sensing, where the usual $\ell_1$ norm on the signal to be recovered is replaced with a $\ell_p$ quasi-norm, where $p < 1$. Similar to our approach, a splitting scheme is used, resulting in a non-convex per-pixel sub-problem. To solve this, a Huber

approximation (see [1]) to the quasi-norm is used, allowing the derivation of a generalized shrinkage operator to solve the sub-problem efficiently. However, this approximates the original sub-problem, unlike our approach.

## 2 Algorithm

We now introduce the non-blind deconvolution problem. $\mathbf{x}$ is the original uncorrupted linear grayscale image of $N$ pixels; $\mathbf{y}$ is an image degraded by blur and/or noise, which we assume to be produced by convolving $\mathbf{x}$ with a blur kernel $\mathbf{k}$ and adding zero mean Gaussian noise. We assume that $\mathbf{y}$ and $\mathbf{k}$ are given and seek to reconstruct $\mathbf{x}$. Given the ill-posed nature of the task, we regularize using a penalty function $|.|^\alpha$ that acts on the output of a set of filters $f_1, \ldots, f_j$ applied to $\mathbf{x}$. A weighting term $\lambda$ controls the strength of the regularization. From a probabilistic perspective, we seek the MAP estimate of $\mathbf{x}$: $p(\mathbf{x}|\mathbf{y}, \mathbf{k}) \propto p(\mathbf{y}|\mathbf{x}, \mathbf{k})p(\mathbf{x})$, the first term being a Gaussian likelihood and second being the hyper-Laplacian image prior. Maximizing $p(\mathbf{x}|\mathbf{y}, \mathbf{k})$ is equivalent to minimizing the cost $-\log p(\mathbf{x}|\mathbf{y}, \mathbf{k})$:

$$\min_{\mathbf{x}} \sum_{i=1}^{N} \left( \frac{\lambda}{2} (\mathbf{x} \oplus \mathbf{k} - \mathbf{y})_i^2 + \sum_{j=1}^{J} |(\mathbf{x} \oplus f_j)_i|^\alpha \right) \tag{1}$$

where $i$ is the pixel index, and $\oplus$ is the 2-dimensional convolution operator. For simplicity, we use two first-order derivative filters $f_1 = [1\ -1]$ and $f_2 = [1\ -1]^T$, although additional ones can easily be added (e.g. learned filters [13, 16], or higher order derivatives). For brevity, we denote $F_i^j \mathbf{x} \equiv (\mathbf{x} \oplus f_j)_i$ for $j = 1, .., J$.

Using the half-quadratic penalty method [5, 6, 22], we now introduce auxiliary variables $w_i^1$ and $w_i^2$ (together denoted as $\mathbf{w}$) at each pixel that allow us to move the $F_i^j \mathbf{x}$ terms outside the $|.|^\alpha$ expression, giving a new cost function:

$$\min_{\mathbf{x}, \mathbf{w}} \sum_i \left( \frac{\lambda}{2} (\mathbf{x} \oplus \mathbf{k} - \mathbf{y})_i^2 + \frac{\beta}{2} \left( \|F_i^1 \mathbf{x} - w_i^1\|_2^2 + \|F_i^2 \mathbf{x} - w_i^2\|_2^2 \right) + |w_i^1|^\alpha + |w_i^2|^\alpha \right) \tag{2}$$

where $\beta$ is a weight that we will vary during the optimization, as described in Section 2.3. As $\beta \to \infty$, the solution of Eqn. 2 converges to that of Eqn. 1. Minimizing Eqn. 2 for a fixed $\beta$ can be performed by alternating between two steps, one where we solve for $\mathbf{x}$, given values of $\mathbf{w}$ and vice-versa. The novel part of our algorithm lies in the $\mathbf{w}$ sub-problem, but first we briefly describe the $\mathbf{x}$ sub-problem and its straightforward solution.

### 2.1 x sub-problem

Given a fixed value of $\mathbf{w}$ from the previous iteration, Eqn. 2 is quadratic in $\mathbf{x}$. The optimal $\mathbf{x}$ is thus:

$$\left( F^{1^T} F^1 + F^{2^T} F^2 + \frac{\lambda}{\beta} K^T K \right) \mathbf{x} = F^{1^T} \mathbf{w}^1 + F^{2^T} \mathbf{w}^2 + \frac{\lambda}{\beta} K^T \mathbf{y} \tag{3}$$

where $K\mathbf{x} \equiv \mathbf{x} \oplus \mathbf{k}$. Assuming circular boundary conditions, we can apply 2D FFT's which diagonalize the convolution matrices $F^1, F^2, K$, enabling us to find the optimal $\mathbf{x}$ directly:

$$\mathbf{x} = \mathcal{F}^{-1} \left( \frac{\mathcal{F}(F^1)^* \circ \mathcal{F}(\mathbf{w}^1) + \mathcal{F}(F^2)^* \circ \mathcal{F}(\mathbf{w}^2) + (\lambda/\beta)\mathcal{F}(K)^* \circ \mathcal{F}(\mathbf{y})}{\mathcal{F}(F^1)^* \circ \mathcal{F}(F^1) + \mathcal{F}(F^2)^* \circ \mathcal{F}(F^2) + (\lambda/\beta)\mathcal{F}(K)^* \circ \mathcal{F}(K)} \right) \tag{4}$$

where $*$ is the complex conjugate and $\circ$ denotes component-wise multiplication. The division is also performed component-wise. Solving Eqn. 4 requires only 3 FFT's at each iteration since many of the terms can be precomputed. The form of this sub-problem is identical to that of [22].

### 2.2 w sub-problem

Given a fixed $\mathbf{x}$, finding the optimal $\mathbf{w}$ consists of solving $2N$ independent 1D problems of the form:

$$w^* = \arg \min_w |w|^\alpha + \frac{\beta}{2} (w - v)^2 \tag{5}$$

where $v \equiv F_i^j \mathbf{x}$. We now describe two approaches to finding $w^*$.

#### 2.2.1 Lookup table

For a fixed value of $\alpha$, $w^*$ in Eqn. 5 only depends on two variables, $\beta$ and $v$, hence can easily be tabulated off-line to form a lookup table. We numerically solve Eqn. 5 for $10,000$ different values of $v$ over the range encountered in our problem ($-0.6 \le v \le 0.6$). This is repeated for different $\beta$ values, namely integer powers of $\sqrt{2}$ between 1 and 256. Although the LUT gives an approximate solution, it allows the $\mathbf{w}$ sub-problem to be solved very quickly for any $\alpha > 0$.

### 2.2.2 Analytic solution

For some specific values of $\alpha$, it is possible to derive exact analytical solutions to the **w** sub-problem. For $\alpha = 2$, the sub-problem is quadratic and thus easily solved. If $\alpha = 1$, Eqn. 5 reduces to a 1-D shrinkage operation [22]. For some special cases of $1 < \alpha < 2$, there exist analytic solutions [26]. Here, we address the more challenging case of $\alpha < 1$ and we now describe a way to solve Eqn. 5 for two special cases of $\alpha = 1/2$ and $\alpha = 2/3$. For non-zero $w$, setting the derivative of Eqn. 5 w.r.t $w$ to zero gives:

$$\alpha |w|^{\alpha-1}\text{sign}(w) + \beta(w - v) = 0 \tag{6}$$

For $\alpha = 1/2$, this becomes, with successive simplification:

$$|w|^{-1/2}\text{sign}(w) + 2\beta(w - v) = 0 \tag{7}$$

$$|w|^{-1} = 4\beta^2(v - w)^2 \tag{8}$$

$$w^3 - 2vw^2 + v^2w - \text{sign}(w)/4\beta^2 = 0 \tag{9}$$

At first sight Eqn. 9 appears to be two different cubic equations with the $\pm 1/4\beta^2$ term, however we need only consider one of these as $v$ is fixed and $w^*$ must lie between $0$ and $v$. Hence we can replace $\text{sign}(w)$ with $\text{sign}(v)$ in Eqn. 9:

$$w^3 - 2vw^2 + v^2w - \text{sign}(v)/4\beta^2 = 0 \tag{10}$$

For the case $\alpha = 2/3$, using a similar derivation, we arrive at:

$$w^4 - 3vw^3 + 3v^2w^2 - v^3w + \frac{8}{27\beta^3} = 0 \tag{11}$$

there being no $\text{sign}(w)$ term as it conveniently cancels in this case. Hence $w^*$, the solution of Eqn. 5, is either $0$ or a root of the cubic polynomial in Eqn. 10 for $\alpha = 1/2$, or equivalently a root of the quartic polynomial in Eqn. 10 for $\alpha = 2/3$. Although it is tempting to try the same manipulation for $\alpha = 3/4$, this results in a 5$^{\text{th}}$ order polynomial, which can only be solved numerically.

**Finding the roots of the cubic and quartic polynomials:** Analytic formulae exist for the roots of cubic and quartic polynomials [23, 24] and they form the basis of our approach, as detailed in Algorithms 2 and 3. In both the cubic and quartic cases, the computational bottleneck is the cube root operation. An alternative way of finding the roots of the polynomials Eqn. 10 and Eqn. 11 is to use a numerical root-finder such as Newton-Raphson. In our experiments, we found Newton-Raphson to be slower and less accurate than either the analytic method or the LUT approach (see [8] for futher details).

**Selecting the correct roots:** Given the roots of the polynomial, we need to determine which one corresponds to the global minima of Eqn. 5. When $\alpha = 1/2$, the resulting cubic equation can have: (a) 3 imaginary roots; (b) 2 imaginary roots and 1 real root, or (c) 3 real roots. In the case of (a), the $|w|^\alpha$ term means Eqn. 5 has positive derivatives around $0$ and the lack of real roots implies the derivative never becomes negative, thus $w^* = 0$. For (b), we need to compare the costs of the single real root and $w = 0$, an operation that can be efficiently performed using Eqn. 13 below. In (c) we have 3 real roots. Examining Eqn. 7 and Eqn. 8, we see that the squaring operation introduces a spurious root above $v$ when $v > 0$, and below $v$ when $v < 0$. This root can be ignored, since $w^*$ must lie between $0$ and $v$. The cost function in Eqn. 5 has a local maximum near $0$ and a local minimum between this local maximum and $v$. Hence of the 2 remaining roots, the one further from $0$ will have a lower cost. Finally, we need to compare the cost of this root with that of $w = 0$ using Eqn. 13.

We can use similar arguments for the $\alpha = 2/3$ case. Here we can potentially have: (a) 4 imaginary roots, (b) 2 imaginary and 2 real roots, or (c) 4 real roots. In (a), $w^* = 0$ is the only solution. For (b), we pick the larger of the 2 real roots and compare the costs with $w = 0$ using Eqn. 13, similar to the case of 3 real roots for the cubic. Case (c) never occurs: the final quartic polynomial Eqn. 11 was derived with a cubing operation from the analytic derivative. This introduces 2 spurious roots into the final solution, both of which are imaginary, thus only cases (a) and (b) are possible.

In both the cubic and quartic cases, we need an efficient way to pick between $w = 0$ and a real root that is between $0$ and $v$. We now describe a direct mechanism for doing this which does not involve the expensive computation of the cost function in Eqn. 5[1].

Let $r$ be the non-zero real root. $0$ must be chosen if it has lower cost in Eqn. 5. This implies:

$$|r|^\alpha + \frac{\beta}{2}(r-v)^2 \quad > \quad \frac{\beta v^2}{2}$$

$$\text{sign}(r)|r|^{\alpha-1} + \frac{\beta}{2}(r-2v) \quad \lessgtr \quad 0 \quad , \quad r \lessgtr 0 \tag{12}$$

Since we are only considering roots of the polynomial, we can use Eqn. 6 to eliminate $\text{sign}(r)|r|^{\alpha-1}$ from Eqn. 6 and Eqn. 12, yielding the condition:

$$r \lessgtr 2v\frac{(\alpha-1)}{(\alpha-2)} \quad , \quad v \gtrless 0 \tag{13}$$

since $\text{sign}(r) = \text{sign}(v)$. So $w^* = r$ if $r$ is between $2v/3$ and $v$ in the $\alpha = 1/2$ case or between $v/2$ and $v$ in the $\alpha = 2/3$ case. Otherwise $w^* = 0$. Using this result, picking $w^*$ can be efficiently coded, e.g. lines 12–16 of Algorithm 2. Overall, the analytic approach is slower than the LUT, but it gives an exact solution to the $\mathbf{w}$ sub-problem.

## 2.3 Summary of algorithm

We now give the overall algorithm using a LUT for the $\mathbf{w}$ sub-problem. As outlined in Algorithm 1 below, we minimize Eqn. 2 by alternating the $\mathbf{x}$ and $\mathbf{w}$ sub-problems $T$ times, before increasing the value of $\beta$ and repeating. Starting with some small value $\beta_0$ we scale it by a factor $\beta_{\text{Inc}}$ until it exceeds some fixed value $\beta_{\text{Max}}$. In practice, we find that a single inner iteration suffices ($T = 1$), although more can sometimes be needed when $\beta$ is small.

---

**Algorithm 1** Fast image deconvolution using hyper-Laplacian priors

---

**Require:** Blurred image $\mathbf{y}$, kernel $\mathbf{k}$, regularization weight $\lambda$, exponent $\alpha$ ($\mathop{\text{¿}}0$)
**Require:** $\beta$ regime parameters: $\beta_0, \beta_{\text{Inc}}, \beta_{\text{Max}}$
**Require:** Number of inner iterations $T$.
 1: $\beta = \beta_0, \mathbf{x} = \mathbf{y}$
 2: Precompute constant terms in Eqn. 4.
 3: **while** $\beta < \beta_{\text{Max}}$ **do**
 4:     $iter = 0$
 5:     **for** $i = 1$ to $T$ **do**
 6:         Given $\mathbf{x}$, solve Eqn. 5 for all pixels using a LUT to give $\mathbf{w}$
 7:         Given $\mathbf{w}$, solve Eqn. 4 to give $\mathbf{x}$
 8:     **end for**
 9:     $\beta = \beta_{\text{Inc}} \cdot \beta$
10: **end while**
11: **return** Deconvolved image $\mathbf{x}$

---

As with any non-convex optimization problem, it is difficult to derive any guarantees regarding the convergence of Algorithm 1. However, we can be sure that the global optimum of each sub-problem will be found, given the fixed $\mathbf{x}$ and $\mathbf{w}$ from the previous iteration. Like other methods that use this form of alternating minimization [5, 6, 22], there is little theoretical guidance for setting the $\beta$ schedule. We find that the simple scheme shown in Algorithm 1 works well to minimize Eqn. 2 and its proxy Eqn. 1. The experiments in Section 3 show our scheme achieves very similar SNR levels to IRLS, but at a greatly lower computational cost.

## 3 Experiments

We evaluate the deconvolution performance of our algorithm on images, comparing them to numerous other methods: (i) $\ell_2$ (Gaussian) prior on image gradients; (ii) Lucy-Richardson [15]; (iii) the algorithm of Wang *et al.* [22] using a total variation (TV) norm prior and (iv) a variant of [22] using an $\ell_1$ (Laplacian) prior; (v) the IRLS approach of Levin *et al.* [10] using a hyper-Laplacian prior with $\alpha = 1/2, 2/3, 4/5$. Note that only IRLS and our method use a prior with $\alpha < 1$. For the IRLS scheme, we used the implementation of [10] with default parameters, the only change being the removal of higher order derivative filters to enable a direct comparison with other approaches. Note that IRLS and $\ell_2$ directly minimize Eqn. 1, while our method, and the TV and $\ell_1$ approaches of [22] minimize the cost in Eqn. 2, using $T = 1, \beta_0 = 1, \beta_{\text{Inc}} = 2\sqrt{2}, \beta_{\text{Max}} = 256$. In our approach, we use $\alpha = 1/2$ and $\alpha = 2/3$, and compare the performance of the LUT and analytic methods as well. All runs were performed with multithreading enabled (over 4 CPU cores).

We evaluate the algorithms using a set of blurry images, created in the following way. 7 in-focus grayscale real-world images were downloaded from the web. They were then blurred by real-world camera shake kernels from [12]. 1% Gaussian noise was added, followed by quantization to 255 discrete values. In any practical deconvolution setting the blur kernel is never perfectly known. Therefore, the kernel passed to the algorithms was a minor perturbation of the true kernel, to mimic kernel estimation errors. In experiments with non-perturbed kernels (not shown), the results are similar to those in Tables 3 and 1 but with slightly higher SNR levels. See Fig. 2 for an example of a kernel from [12] and its perturbed version. Our evaluation metric was the SNR between the original image $\hat{\mathbf{x}}$ and the deconvolved output $\mathbf{x}$, defined as $10\log_{10}\frac{\|\hat{\mathbf{x}}-\mu(\hat{\mathbf{x}})\|^2}{\|\hat{\mathbf{x}}-\mathbf{x}\|^2}$, $\mu(\hat{\mathbf{x}})$ being the mean of $\hat{\mathbf{x}}$.

In Table 1 we compare the algorithms on 7 different images, all blurred with the same $19\times19$ kernel. For each algorithm we exhaustively searched over different regularization weights $\lambda$ to find the value that gave the best SNR performance, as reported in the table. In Table 3 we evaluate the algorithms with the same $512\times512$ image blurred by 8 different kernels (from [12]) of varying size. Again, the optimal value of $\lambda$ for each kernel/algorithm combination was chosen from a range of values based on SNR performance. Table 2 shows the running time of several algorithms on images up to $3072\times3072$ pixels. Figure 2 shows a larger $27\times27$ blur being deconvolved from two example images, comparing the output of different methods.

The tables and figures show our method with $\alpha = 2/3$ and IRLS with $\alpha = 4/5$ yielding higher quality results than other methods. However, our algorithm is around 70 to 350 times faster than IRLS depending on whether the analytic or LUT method is used. This speedup factor is independent of image size, as shown by Table 2. The $\ell_1$ method of [22] is the best of the other methods, being of comparable speed to ours but achieving lower SNR scores. The SNR results for our method are almost the same whether we use LUTs or analytic approach. Hence, in practice, the LUT method is preferred, since it is approximately 5 times faster than the analytic method and can be used for any value of $\alpha$.

| Image # | Blurry | $\ell_2$ | Lucy | TV | $\ell_1$ | IRLS $\alpha=1/2$ | IRLS $\alpha=2/3$ | IRLS $\alpha=4/5$ | Ours $\alpha=1/2$ | Ours $\alpha=2/3$ |
|---|---|---|---|---|---|---|---|---|---|---|
| 1 | 6.42 | 14.13 | 12.54 | 15.87 | 16.18 | 14.61 | 15.45 | 16.04 | 16.05 | **16.44** |
| 2 | 10.73 | 17.56 | 15.15 | 19.37 | 19.86 | 18.43 | 19.37 | 20.00 | 19.78 | **20.26** |
| 3 | 12.45 | 19.30 | 16.68 | 21.83 | 22.77 | 21.53 | 22.62 | 22.95 | 23.26 | **23.27** |
| 4 | 8.51 | 16.02 | 14.27 | 17.66 | 18.02 | 16.34 | 17.31 | 17.98 | 17.70 | **18.17** |
| 5 | 12.74 | 16.59 | 13.28 | 19.34 | 20.25 | 19.12 | 19.99 | 20.20 | **21.28** | 21.00 |
| 6 | 10.85 | 15.46 | 12.00 | 17.13 | 17.59 | 15.59 | 16.58 | 17.04 | 17.79 | **17.89** |
| 7 | 11.76 | 17.40 | 15.22 | 18.58 | 18.85 | 17.08 | 17.99 | 18.61 | 18.58 | **18.96** |
| Av. SNR gain | | 6.14 | 3.67 | 8.05 | 8.58 | 7.03 | 7.98 | 8.48 | 8.71 | **8.93** |
| Av. Time (secs) | | 79.85 | 1.55 | 0.66 | 0.75 | 354 | 354 | 354 | L:1.01 A:5.27 | L:1.00 A:4.08 |

Table 1: Comparison of SNRs and running time of 9 different methods for the deconvolution of 7 $576\times864$ images, blurred with the same $19\times19$ kernel. L=Lookup table, A=Analytic. The best performing algorithm for each kernel is shown in bold. Our algorithm with $\alpha = 2/3$ beats IRLS with $\alpha = 4/5$, as well as being much faster. On average, both these methods outperform $\ell_1$, demonstrating the benefits of a sparse prior.

| Image size | $\ell_1$ | IRLS $\alpha=4/5$ | Ours (LUT) $\alpha=2/3$ | Ours (Analytic) $\alpha=2/3$ |
|---|---|---|---|---|
| $256\times256$ | 0.24 | 78.14 | 0.42 | 0.7 |
| $512\times512$ | 0.47 | 256.87 | 0.55 | 2.28 |
| $1024\times1024$ | 2.34 | 1281.3 | 2.78 | 10.87 |
| $2048\times2048$ | 9.34 | 4935 | 10.72 | 44.64 |
| $3072\times3072$ | 22.40 | - | 24.07 | 100.42 |

Table 2: Run-times of different methods for a range of image sizes, using a $13\times13$ kernel. Our LUT algorithm is more than 100 times faster than the IRLS method of [10].

## 4 Discussion

We have described an image deconvolution scheme that is fast, conceptually simple and yields high quality results. Our algorithm takes a novel approach to the non-convex optimization prob-

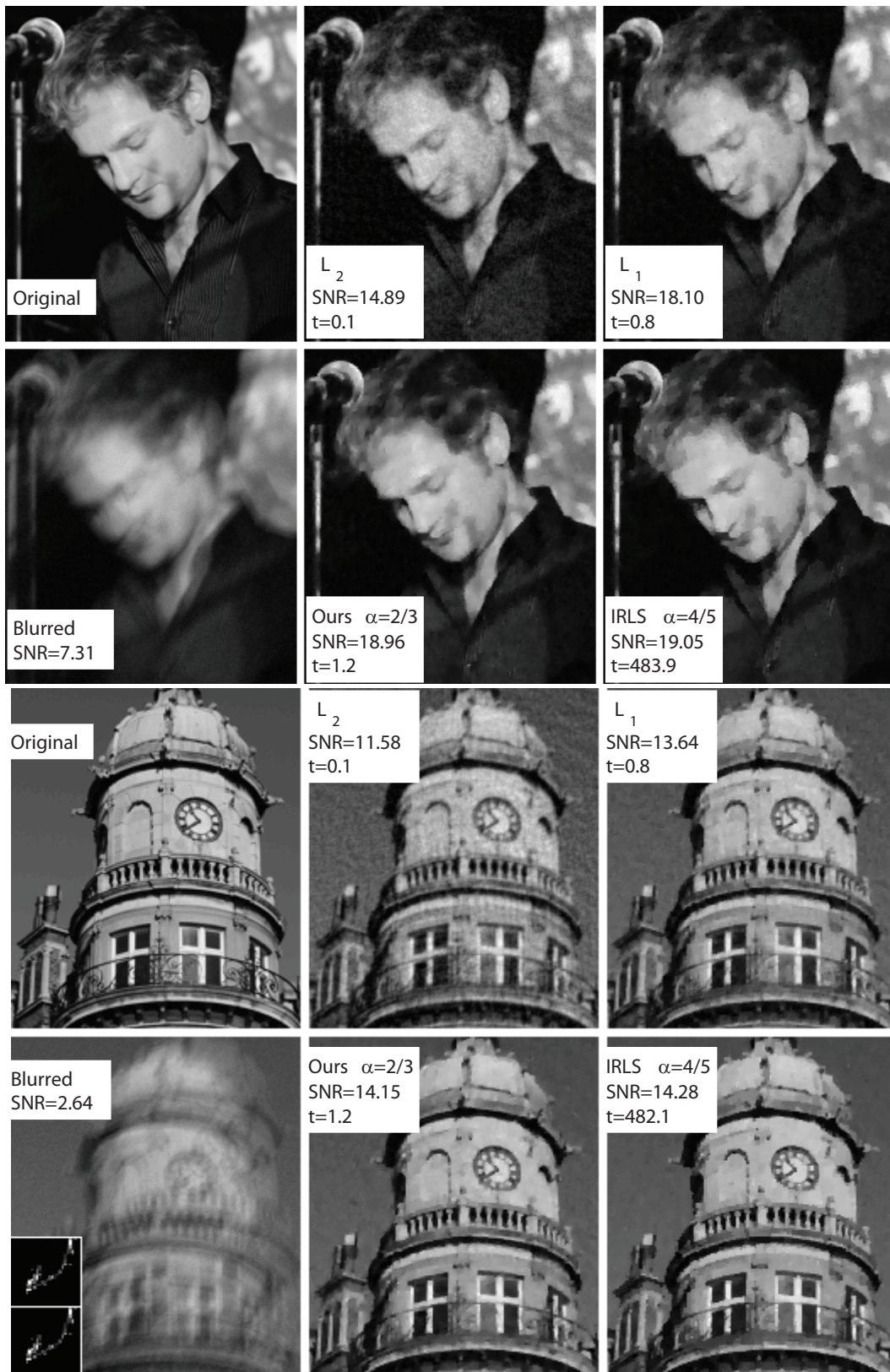

Figure 2: Crops from two images (#1 & #5) being deconvolved by 4 different algorithms, including ours using a 27×27 kernel (#7). In the bottom left inset, we show the original kernel from [12] (lower) and the perturbed version provided to the algorithms (upper), to make the problem more realistic. This figure is best viewed on screen, rather than in print.

| Kernel # / size | Blurry | $\ell_2$ | Lucy | TV | $\ell_1$ | IRLS $\alpha$=1/2 | IRLS $\alpha$=2/3 | IRLS $\alpha$=4/5 | Ours $\alpha$=1/2 | Ours $\alpha$=2/3 |
|---|---|---|---|---|---|---|---|---|---|---|
| #1: 13×13 | 10.69 | 17.22 | 14.49 | 19.21 | 19.41 | 17.20 | 18.22 | 18.87 | 19.36 | **19.66** |
| #2: 15×15 | 11.28 | 16.14 | 13.81 | 17.94 | 18.29 | 16.17 | 17.26 | 18.02 | 18.14 | **18.64** |
| #3: 17×17 | 8.93 | 14.94 | 12.16 | 16.50 | 16.86 | 15.34 | 16.36 | 16.99 | 16.73 | **17.25** |
| #4: 19×19 | 10.13 | 15.27 | 12.38 | 16.83 | 17.25 | 15.97 | 16.98 | 17.57 | 17.29 | **17.67** |
| #5: 21×21 | 9.26 | 16.55 | 13.60 | 18.72 | 18.83 | 17.23 | 18.36 | 18.88 | 19.11 | **19.34** |
| #6: 23×23 | 7.87 | 15.40 | 13.32 | 17.01 | 17.42 | 15.66 | 16.73 | 17.40 | 17.26 | **17.77** |
| #7: 27×27 | 6.76 | 13.81 | 11.55 | 15.42 | 15.69 | 14.59 | 15.68 | **16.38** | 15.92 | 16.29 |
| #8: 41×41 | 6.00 | 12.80 | 11.19 | 13.53 | 13.62 | 12.68 | 13.60 | **14.25** | 13.73 | 13.68 |
| Av. SNR gain | | 6.40 | 3.95 | 8.03 | 8.31 | 6.74 | 7.78 | 8.43 | 8.33 | **8.67** |
| Av. Time (sec) | | 57.44 | 1.22 | 0.50 | 0.55 | 271 | 271 | 271 | L:0.81 A:2.15 | L:0.78 A:2.23 |

Table 3: Comparison of SNRs and running time of 9 different methods for the deconvolution of a 512×512 image blurred by 7 different kernels. L=Lookup table, A=Analytic. Our algorithm beats all other methods in terms of quality, with the exception of IRLS on the largest kernel size. However, our algorithm is far faster than IRLS, being comparable in speed to the $\ell_1$ approach.

lem arising from the use of a hyper-Laplacian prior, by using a splitting approach that allows the non-convexity to become separable over pixels. Using a LUT to solve this sub-problem allows for orders of magnitude speedup in the solution over existing methods. Our Matlab implementation is available online at http://cs.nyu.edu/~dilip/wordpress/?page_id=122.

A potential drawback to our method, common to the TV and $\ell_1$ approaches of [22], is its use of frequency domain operations which assume circular boundary conditions, something not present in real images. These give rise to boundary artifacts which can be overcome to some extend with edge tapering operations. However, our algorithm is suitable for very large images where the boundaries are a small fraction of the overall image.

Although we focus on deconvolution, our scheme can be adapted to a range of other problems which rely on natural image statistics. For example, by setting $\mathbf{k} = 1$ the algorithm can be used to denoise, or if $\mathbf{k}$ is a defocus kernel it can be used for super-resolution. The speed offered by our algorithm makes it practical to perform these operations on the multi-megapixel images from modern cameras.

---

**Algorithm 2:** Solve Eqn. 5 for $\alpha = 1/2$

**Require:** Target value $v$, Weight $\beta$
1: $\epsilon = 10^{-6}$
2: {Compute intermediary terms $m, t_1, t_2, t_3$}
3: $m = -\text{sign}(v)/4\beta^2$
4: $t_1 = 2v/3$
5: $t_2 = \sqrt[3]{-27m - 2v^3 + 3\sqrt{3}\sqrt{27m^2 + 4mv^3}}$
6: $t_3 = v^2/t_2$
7: {Compute 3 roots, $r_1, r_2, r_3$:}
8: $r_1 = t_1 + 1/(3 \cdot 2^{1/3}) \cdot t_2 + 2^{1/3}/3 \cdot t_3$
9: $r_2 = t_1 - (1 - \sqrt{3}i)/(6 \cdot 2^{1/3}) \cdot t_2$
   $\quad - (1 + \sqrt{3}i)/(3 \cdot 2^{2/3}) \cdot t_3$
10: $r_3 = t_1 - (1 + \sqrt{3}i)/(6 \cdot 2^{1/3}) \cdot t_2$
   $\quad - (1 - \sqrt{3}i)/(3 \cdot 2^{2/3}) \cdot t_3$
11: {Pick global minimum from $(0, r_1, r_2, r_3)$}
12: $r = [r_1, r_2, r_3]$
13: $c_1 = (\text{abs}(\text{imag}(r)) < \epsilon)$ {Root must be real}
14: $c_2 = \text{real}(r)\text{sign}(v) > (2/3 \cdot \text{abs}(v))$
   {Root must obey bound of Eqn. 13}
15: $c_3 = \text{real}(r)\text{sign}(v) < \text{abs}(v)$ {Root $< v$}
16: $w^* = \max((c_1 \& c_2 \& c_3)\text{real}(r)\text{sign}(v))\text{sign}(v)$
**return** $w^*$

---

**Algorithm 3:** Solve Eqn. 5 for $\alpha = 2/3$

**Require:** Target value $v$, Weight $\beta$
1: $\epsilon = 10^{-6}$
2: {Compute intermediary terms $m, t_1, \ldots, t_7$:}
3: $m = 8/(27\beta^3)$
4: $t_1 = -9/8 \cdot v^2$
5: $t_2 = v^3/4$
6: $t_3 = -1/8 \cdot mv^2$
7: $t_4 = -t_3/2 + \sqrt{-m^3/27 + m^2v^4/256}$
8: $t_5 = \sqrt[3]{t_4}$
9: $t_6 = 2(-5/18 \cdot t_1 + t_5 + m/(3 \cdot t_5))$
10: $t_7 = \sqrt{t_1/3 + t_6}$
11: {Compute 4 roots, $r_1, r_2, r_3, r_4$:}
12: $r_1 = 3v/4 + (t_7 + \sqrt{-(t_1 + t_6 + t_2/t_7)})/2$
13: $r_2 = 3v/4 + (t_7 - \sqrt{-(t_1 + t_6 + t_2/t_7)})/2$
14: $r_3 = 3v/4 + (-t_7 + \sqrt{-(t_1 + t_6 - t_2/t_7)})/2$
15: $r_4 = 3v/4 + (-t_7 - \sqrt{-(t_1 + t_6 - t_2/t_7)})/2$
16: {Pick global minimum from $(0, r_1, r_2, r_3, r_4)$}
17: $r = [r_1, r_2, r_3, r_4]$
18: $c_1 = (\text{abs}(\text{imag}(r)) < \epsilon)$ {Root must be real}
19: $c_2 = \text{real}(r)\text{sign}(v) > (1/2 \cdot \text{abs}(v))$
   {Root must obey bound in Eqn. 13}
20: $c_3 = \text{real}(r)\text{sign}(v) < \text{abs}(v)$ {Root $< v$}
21: $w^* = \max((c_1 \& c_2 \& c_3)\text{real}(r)\text{sign}(v))\text{sign}(v)$
**return** $w^*$

## Footnotes

[1]This requires the calculation of a fractional power, which is slow, particularly if $\alpha = 2/3$.

# References

[1] R. Chartrand. Fast algorithms for nonconvex compressive sensing: Mri reconstruction from very few data. In *IEEE International Symposium on Biomedical Imaging (ISBI)*, 2009.

[2] R. Chartrand and V. Staneva. Restricted isometry properties and nonconvex compressive sensing. *Inverse Problems*, 24:1–14, 2008.

[3] R. Fergus, B. Singh, A. Hertzmann, S. T. Roweis, and W. Freeman. Removing camera shake from a single photograph. *ACM TOG (Proc. SIGGRAPH)*, 25:787–794, 2006.

[4] D. Field. What is the goal of sensory coding? *Neural Computation*, 6:559–601, 1994.

[5] D. Geman and G. Reynolds. Constrained restoration and recovery of discontinuities. *PAMI*, 14(3):367–383, 1992.

[6] D. Geman and C. Yang. Nonlinear image recovery with half-quadratic regularization. *PAMI*, 4:932–946, 1995.

[7] N. Joshi, L. Zitnick, R. Szeliski, and D. Kriegman. Image deblurring and denoising using color priors. In *CVPR*, 2009.

[8] D. Krishnan and R. Fergus. Fast image deconvolution using hyper-laplacian priors, supplementary material. *NYU Tech. Rep. 2009*, 2009.

[9] A. Levin. Blind motion deblurring using image statistics. In *NIPS*, 2006.

[10] A. Levin, R. Fergus, F. Durand, and W. Freeman. Image and depth from a conventional camera with a coded aperture. *ACM TOG (Proc. SIGGRAPH)*, 26(3):70, 2007.

[11] A. Levin and Y. Weiss. User assisted separation of reflections from a single image using a sparsity prior. *PAMI*, 29(9):1647–1654, Sept 2007.

[12] A. Levin, Y. Weiss, F. Durand, and W. T. Freeman. Understanding and evaluating blind deconvolution algorithms. In *CVPR*, 2009.

[13] S. Osindero, M. Welling, and G. Hinton. Topographic product models applied to natural scene statistics. *Neural Computation*, 1995.

[14] J. Portilla, V. Strela, M. J. Wainwright, and E. P. Simoncelli. Image denoising using a scale mixture of Gaussians in the wavelet domain. *IEEE TIP*, 12(11):1338–1351, November 2003.

[15] W. Richardson. Bayesian-based iterative method of image restoration. 62:55–59, 1972.

[16] S. Roth and M. J. Black. Fields of Experts: A Framework for Learning Image Priors. In *CVPR*, volume 2, pages 860–867, 2005.

[17] L. Rudin, S. Osher, and E. Fatemi. Nonlinear total variation based noise removal algorithms. *Physica D*, 60:259–268, 1992.

[18] E. Simoncelli and E. H. Adelson. Noise removal via bayesian wavelet coring. In *ICIP*, pages 379–382, 1996.

[19] C. V. Stewart. Robust parameter estimation in computer vision. *SIAM Reviews*, 41(3):513–537, Sept. 1999.

[20] M. F. Tappen, B. C. Russell, and W. T. Freeman. Exploiting the sparse derivative prior for super-resolution and image demosaicing. In *SCTV*, 2003.

[21] M. Wainwright and S. Simoncelli. Scale mixtures of gaussians and teh statistics of natural images. In *NIPS*, pages 855–861, 1999.

[22] Y. Wang, J. Yang, W. Yin, and Y. Zhang. A new alternating minimization algorithm for total variation image reconstruction. *SIAM J. Imaging Sciences*, 1(3):248–272, 2008.

[23] E. W. Weisstein. Cubic formula. http://mathworld.wolfram.com/CubicFormula.html.

[24] E. W. Weisstein. Quartic equation. http://mathworld.wolfram.com/QuarticEquation.html.

[25] M. Welling, G. Hinton, and S. Osindero. Learning sparse topographic representations with products of student-t distributions. In *NIPS*, 2002.

[26] S. Wright, R. Nowak, and M. Figueredo. Sparse reconstruction by separable approximation. *IEEE Trans. Signal Processing*, page To appear, 2009.

